# Neural Network Analysis of Event Related Potentials and Electroencephalogram Predicts Vigilance

Rita Venturini                    William W. Lytton

Terrence J. Sejnowski
Computational Neurobiology Laboratory
The Salk Institute
La Jolla, CA 92037

## Abstract

Automated monitoring of vigilance in attention intensive tasks such as air traffic control or sonar operation is highly desirable. As the operator monitors the instrument, the instrument would monitor the operator, insuring against lapses. We have taken a first step toward this goal by using feedforward neural networks trained with backpropagation to interpret event related potentials (ERPs) and electroencephalogram (EEG) associated with periods of high and low vigilance. The accuracy of our system on an ERP data set averaged over 28 minutes was 96%, better than the 83% accuracy obtained using linear discriminant analysis. Practical vigilance monitoring will require prediction over shorter time periods. We were able to average the ERP over as little as 2 minutes and still get 90% correct prediction of a vigilance measure. Additionally, we achieved similarly good performance using segments of EEG power spectrum as short as 56 sec.

## 1   INTRODUCTION

Many tasks in society demand sustained attention to minimally varying stimuli over a long period of time. Detection of failure in vigilance during such tasks would be of enormous value. Different physiological variables like electroencephalogram

(EEG), electro-oculogram (EOG), heart rate, and pulse correlate to some extent with the level of attention (1, 2, 3). Profound changes in the appearance and spectrum of the EEG with sleep and drowsiness are well known. However, there is no agreement as to which EEG bands can best predict changes in vigilance. Recent studies (4) seem to indicate that there is a strong correlation between several EEG power spectra frequencies changes and attentional level in subjects performing a sustained task. Another measure that has been widely assessed in this context involves the use of event-related potentials (ERP)(5). These are voltage changes in the ongoing EEG that are time locked to sensory, motor, or cognitive events. They are usually too small to be recognized in the background electrical activity. The ERP's signal is typically extracted from the background noise of the EEG as a consequence of averaging over many trials. The ERP waveform remains constant for each repetition of the event, whereas the background EEG activity has random amplitude. Late cognitive event-related potentials, like the P300, are well known to be related to attentional allocation (6, 7, 8). Unfortunately, these ERPs are evoked only when the subject is attending to a stimulus. This condition is not present in a monitoring situation where monitoring is done precisely because the time of stimulus occurrence is unknown. Instead, shorter latency responses, evoked from unobtrusive task-irrelevant signals, need to be evaluated.

Data from a sonar simulation task was obtained from S.Makeig at al (9). They presented auditory targets only slightly louder than background noise to 13 male United States Navy personnel. Other tones, which the subjects were instructed to ignore, appeared randomly every 2-4 seconds (task irrelevant probes). Background EEG and ERP were both collected and analyzed. The ERPs evoked by the task irrelevant probes were classified into two groups depending on whether they appeared before a correctly identified target (pre-hit ERPs) or a missed target (pre-lapse ERPs). Pre-lapse ERPs showed a relative increase of P2 and N2 components and a decrease of the N1 deflection. N1, N2 and P2 designate the sign and time of peak of components in the ERP. Prior linear discriminant analysis (LDA) performed on the averages of each session, showed 83% correct classification using ERPs obtained from a single scalp site. Thus, the pre-hit and pre-lapse ERPs differed enough to permit classification by averaging over a large enough sample. In addition, EEG power spectra over 81 frequency bands were computed. EEG classification was made on the basis of a continuous measure of performance, the error rate, calculated as the mean of hits and lapses in a 32 sec moving window. Analysis of the EEG power spectrum (9) revealed that significant coherence is observed between various EEG frequencies and performance.

## 2   METHOD

### 2.1   THE DATA SET

Two different groups of input data were used (ERPs and EEG). For the former, a 600 msec sample of task irrelevant probe ERP was reduced to 40 points after low-pass filtering. We normalized the data on the basis of the maximum and minimum values of the entire set, maintaining amplitude variability. A single ERP was classified as being pre-hit or pre-lapse based on the subject's performance on the next target tone. EEG power spectrum, obtained every 1.6 seconds, was used as an input to

predict a continuous estimate of vigilance (error rate), obtained by averaging the subject's performance during a 32 second window (normalized between -1 and 1). The five frequencies used (3, 10, 13, 19 and 39 Hz) had previously shown to be most strongly related to error rate changes (9). Each frequency was individually normalized to range between -1 and 1.

## 2.2 THE NETWORK

Feedforward networks were trained with backpropagation. We compared two-layer network to three-layer networks, varying the number of hidden units in different simulations between 2 and 8. Each architecture was trained ten times on the same task, resetting the weights every time with a different random seed. Initial simulations were performed to select network parameter values. We used a learning rate of 0.3 divided by the fan-in and weight initialization in a range between ±0.3. For the ERP data we used a jackknife procedure. For each simulation, a single pattern was excluded from the training set and considered to be the test pattern. Each pattern in turn was removed and used as the test pattern while the others are used for training. The EEG data set was not as limited as the ERP one and the simulations were performed using half of the data as training and the remaining part as testing set. Therefore, for subjects that had two runs each, the training and testing data came from separate sessions.

## 3  RESULTS

### 3.1  ERPs

The first simulation was done using a two-layer network to assess the adequacy of the neural network approach relative to the previous LDA results. The data set consisted of the grand averages of pre-hits and pre-lapses, from a single scalp site (Cz) of 9 subjects, three of them with a double session, giving a total of 24 patterns. The jackknife procedure was done in two different ways. First each ERP was considered individually, as had been done in the LDA study (pattern-jackknife). Second all the ERPs of a single subject were grouped together and removed together to form the test set (subject-jackknife). The network was trained for 10,000 epochs before testing. Figure 1 shows the weights for the 24 networks each trained with a set of ERPs obtained by removing a single ERP. The "waveform" of the weight values corresponds to features common to the pre-hit ERPs and to the negative of features common to the pre-lapse ERPs. Classification of patterns by the network was considerably more accurate than the 83% correct that had been obtained with the previous LDA analysis. 96% correct evaluation was seen in seven of the ten networks started with different random weight selections. The remaining three networks produced 92% correct responses (Fig. 2). The same two patterns were missed in all cases. Using hidden units did not improve generalization. The subject-jackknife results were very similar: 96% correct in two of ten networks and 92% in the remaining eight (Fig. 2). Thus, there was a somewhat increased difficulty in generalizing across individuals. The ability of the network to generalize over a shorter period of time was tested by progressively decreasing the number of trials used for testing using a network trained on the grand average ERPs. Subaverages

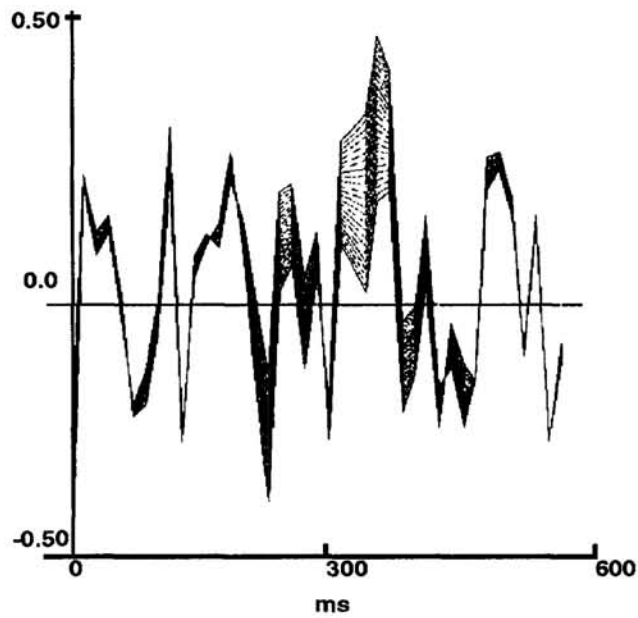

Figure 1: Weights from 24 two-layer networks trained from different initial weights: each value correspond to a sample point in time in the input data.

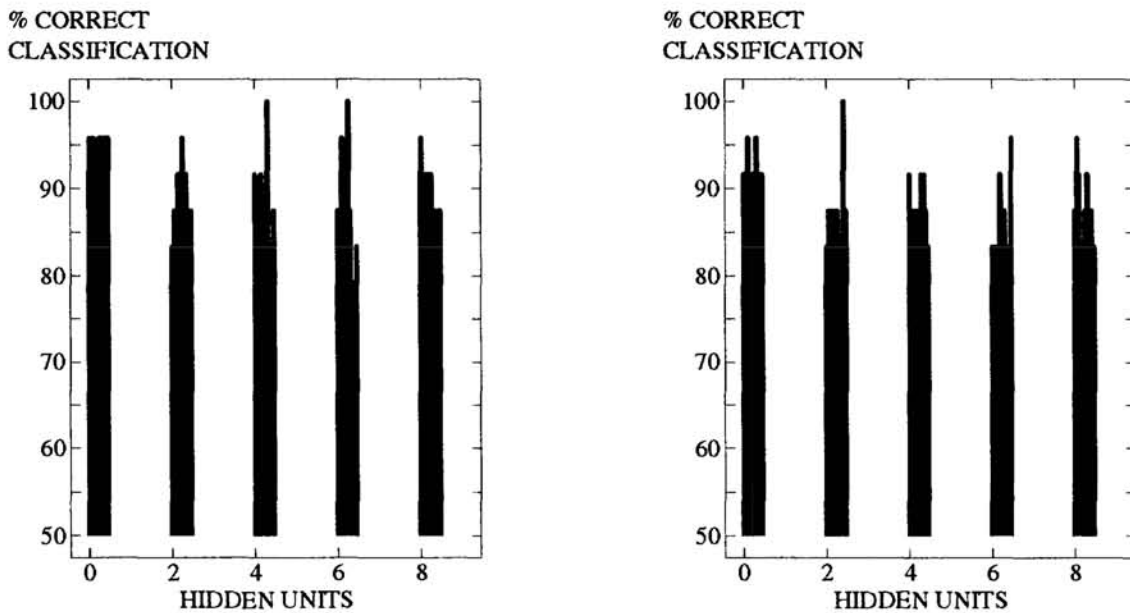

Figure 2: Generalization performance in Pattern (left) and Subject (right) Jack-knifes, using two-layer and three-layer networks with different number of hidden units. Each bar represents a different random start of the network.

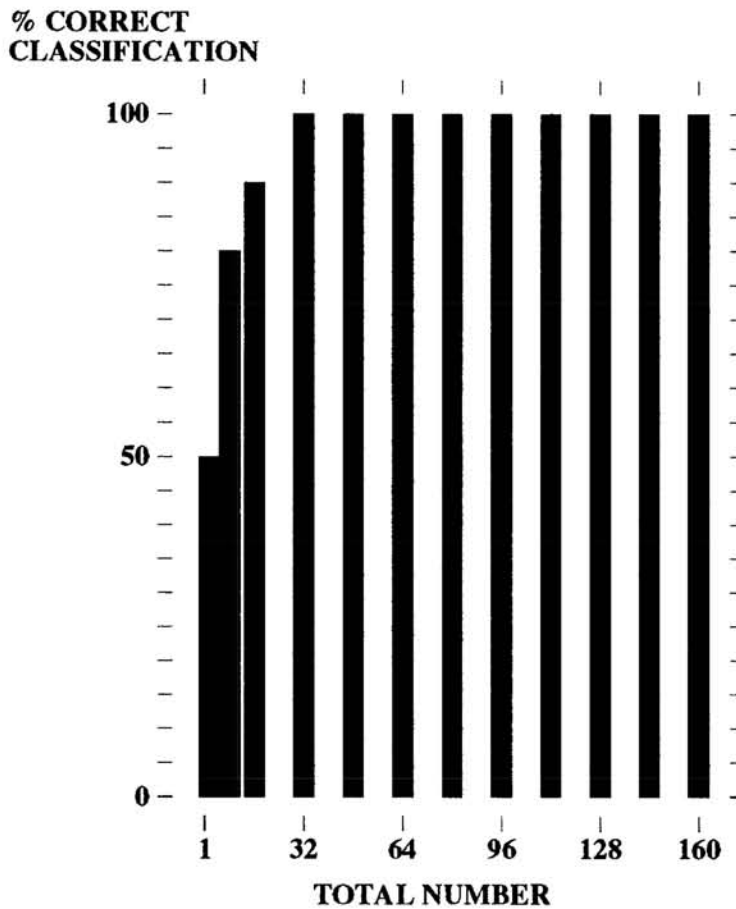

Figure 3: Generalization for testing subaverages made using varying number of individual ERPs

were formed using from 1 to 160 individual ERPs (Figure 3). Performance with a single ERP is at chance. With 16 ERPs, corresponding to about 2 minutes, 90% accuracy was obtained.

## 3.2    EEG

We first report results using a two-layer network to compare with the previous LDA analysis. Five power spectrum frequency bands from a single scalp site (Cz) were used as input data. The error rate was averaged over 32 seconds at 1.6 second intervals. In the first set of runs both error rate and power spectra were filtered using a two minute time window. Good results could be obtained in cases where a subject made errors more than 40% of the time (Fig. 4). When the subject made few errors, training was more difficult and generalization was poor. These results were virtually identical to the LDA ones. The lack of improvement is probably due to the fact that the LDA performance was already close to 90% on this data set. Use of three-layer networks did not improve the generalization performance.

The use of a running average includes information in the EEG after the time at which the network is making a prediction. Causal prediction was attempted using multiple power spectra taken at 1.6 sec intervals over the past 56 sec, to predict the upcoming error rate. The results for one subject are shown in Figure 5. The predicted error rate differs from the target with a root mean square error of 0.3.

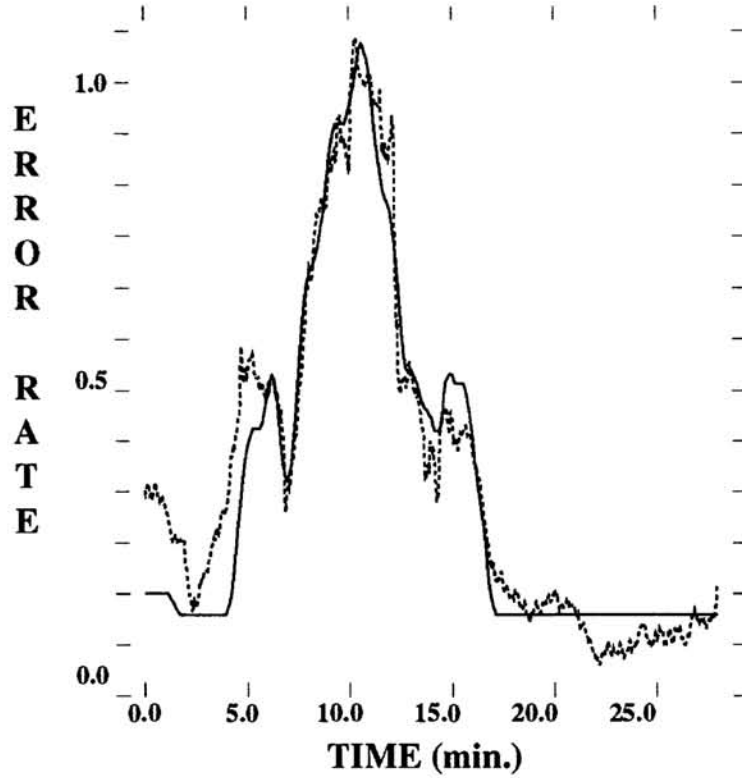

Figure 4: Generalization results predicting error rate from EEG. The dotted line is the network output, solid line the desired value.

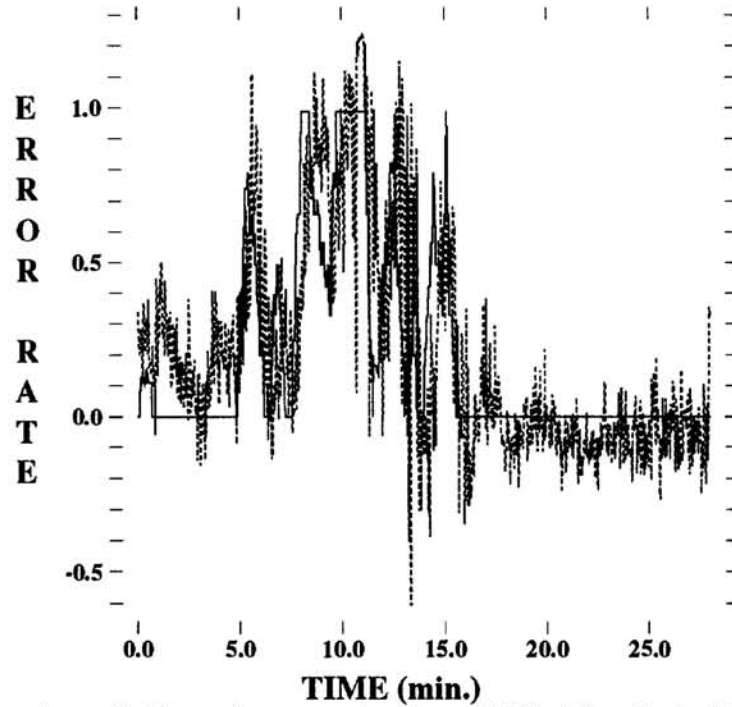

Figure 5: Causal prediction of error rate from EEG. The dotted line is the network output, solid line the desired value.

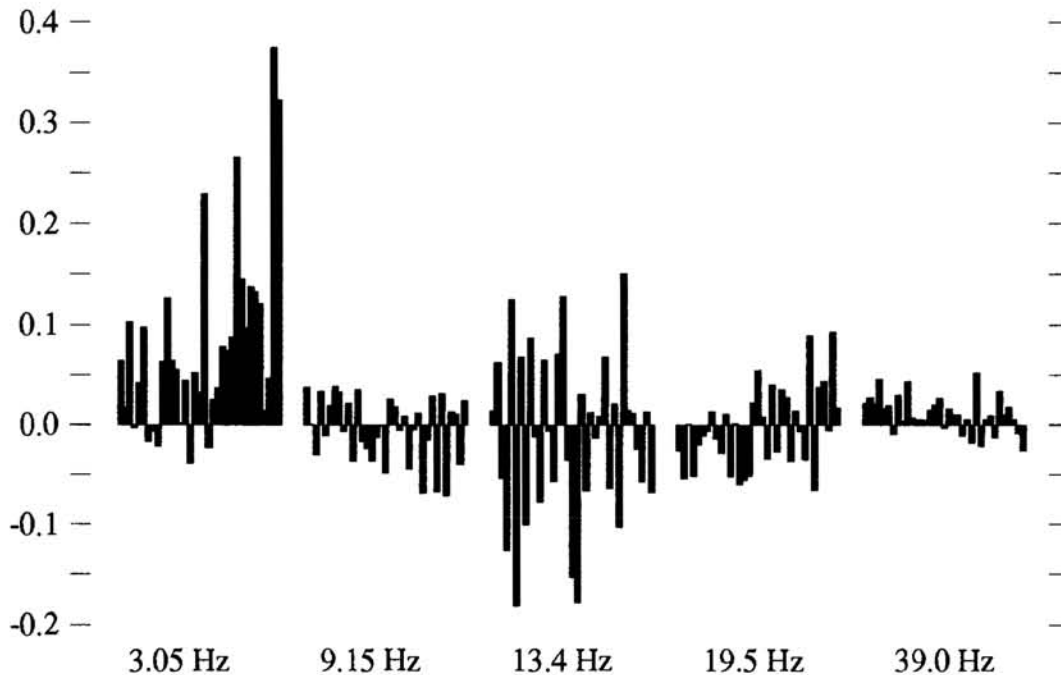

Figure 6: Weights from a two-layer causal prediction network. Each bar, within each frequency band, represents the influence on the output unit of power in that band at previous times ranging from 1 sec (right bar) to 56 sec (left bar).

Figure 6 shows the weights from a two-layer network trained to predict instantaneous error rate. The network mostly uses information from the 3.05 Hz and 13.4 Hz frequency bands in predicting the error rate changes. The values of the 3.05 Hz weights have a strong peak from the most recent time steps, indicating that power in this frequency band predicts the state of vigilance on a short time scale. The alternating positive and negative weights present in the 13.4 Hz set suggest that rapid changes in power in this band might be predictive of vigilance (i.e. the derivative of the power signal).

## 4  DISCUSSION

These results indicate that neural networks could be useful in analyzing electro-physiological measures. The EEG results suggest that the analysis can be applied to detect fluctuations of the attentional level of the subjects in real time. EEG analysis could also be a useful tool for understanding changes that occur in the electric activity of the brain during different states of attention.

In the ERP analysis, the lack of improvement with the introduction of hidden units might be due to the small size of the data set. If the data set is too small, adding hidden units and connections may reduce the ability to find a general solution to the problem. The ERP subject-jackknife results point out that inter-subject generalization is possible. This suggests the possibility of preparing a pre-programmed network that could be used with multiple subjects rather than training the network for each individual. The subaverages results suggest that the detection is possible

in a relatively brief time interval. ERPs could be an useful completion to the EEG analysis in order to obtain an on line detector of attentional changes.

Future research will combination of these two measures along with EOG and heart rate. The idea is to let the model choose different network architectures and parameters, depending on the specific subtask.

## ACKNOWLEDGEMENTS

We would like to thank Scott Makeig and Mark Inlow, Cognitive Performance and Psychophysiology Department, Naval Health Research Center, San Diego for providing the data and for invaluable discussions and Y. Le Cun and L.Y. Bottou from Neuristique who provided the SN2 simulator. RV was supported by Ministry of Public Instruction, Italy; WWL from a Physician Scientist Award, National Institute of Aging; TJS is an Investigator with the Howard Hughes Medical Institute. Research was supported by ONR Grant N00014-91-J-1674.

## REFERENCES

1 Belyavin, A. and Wright, N.A.(1987). Changes in electrical activity of the brain with vigilance. *Electroencephalography and Clinical Neuroscience*, 66:137-144.

2 Torsvall, L. and Akerstedt, T.(1988). Extreme sleepiness: quantification of OG and spectral EEG parameters. *Int. J. Neuroscience*, 38:435-441.

3 Fruhstorfer, H., Langanke, P., Meinzer, K., Peter, J.H., and Pfaff, U.(1977). Neurophysiological vigilance indicators and operational analysis of a train vigilance monitoring device: a laboratory and field study. In R.R.Mackie(Ed.), *Vigilance Theory, Operational Performance, and Physiological Correlates*, 147-162, New York: Plenum Press.

4 Makeig, S. and Inlow M.(1991). Lapses in Alertness : Coherence of fluctuations in performance and EEG spectrum. Cognitive Performance and Psychophysiology Department, NHRC, San Diego. Technical Report.

5 Fruhstorfer, H. and Bergstrom, R.M.(1969). Human vigilance and auditory evoked responses. *Electroencephalography and Clinical Neurophysiology*, 27:346-355.

6 Polich, J.(1989). Habituation of P300 from auditory stimuli. *Psychobiology*, 17:19-28.

7 Polich, J.(1987). Task difficulty, probability, and inter-stimulus interval as determinants of P300 from auditory stimuli. *Electroencephalography and clinical Neurophysiology*, 68:311-320.

8 Polich, J.(1990). P300, Probability, and Interstimulus Interval. *Psychophysiology*, 27:396-403.

9 Makeig S., Elliot F.S., Inlow M. and Kobus D.A.(1991) Predicting Lapses in Vigilance Using Brain Evoked Responses to Irrelevant Auditory Probe. Cognitive Performance and Psychophysiology Department, NHRC, San Diego. Technical Report.